# Robustness and risk-sensitivity in Markov decision processes

**Takayuki Osogami**
IBM Research - Tokyo
5-6-52 Toyosu, Koto-ku, Tokyo, Japan
osogami@jp.ibm.com

## Abstract

We uncover relations between robust MDPs and risk-sensitive MDPs. The objective of a robust MDP is to minimize a function, such as the expectation of cumulative cost, for the worst case when the parameters have uncertainties. The objective of a risk-sensitive MDP is to minimize a risk measure of the cumulative cost when the parameters are known. We show that a risk-sensitive MDP of minimizing the expected exponential utility is equivalent to a robust MDP of minimizing the worst-case expectation with a penalty for the deviation of the uncertain parameters from their nominal values, which is measured with the Kullback-Leibler divergence. We also show that a risk-sensitive MDP of minimizing an iterated risk measure that is composed of certain coherent risk measures is equivalent to a robust MDP of minimizing the worst-case expectation when the possible deviations of uncertain parameters from their nominal values are characterized with a concave function.

## 1 Introduction

Robustness against uncertainties and sensitivity to risk are major issues that have been addressed in recent development of the Markov decision process (MDP). The robust MDP [3, 4, 10, 11, 12, 20, 21] deals with uncertainties in parameters; that is, some of the parameters of the MDP are not known exactly. The objective of a robust MDP is to minimize a function for the worst case when the values of its parameters vary within a predefined set called an uncertainty set. The standard objective function is the expected cumulative cost [11]. When the uncertainty set is trivial, the robust MDP is reduced to the standard MDP [17]. The risk-sensitive MDP [5, 7, 13, 14, 19], on the other hand, assumes that the parameters are exactly known. The objective of a risk-sensitive MDP is to minimize the value of a risk measure, such as the expected exponential utility [5, 7, 8, 15, 18], of the cumulative cost. When the risk measure is expectation, the risk-sensitive MDP is reduced to the standard MDP. The robust MDP and the risk-sensitive MDP have been developed independently.

The goal of this paper is to reveal relations between these two seemingly unrelated models of MDPs. Such unveiled relations will provide insights into the two models of MDPs. For example, it is not always clear what it means to minimize the value of a risk measure or to minimize the worst case expected cumulative cost under an uncertainty set. In particular, the iterated risk measure studied in [13, 14, 19] is defined recursively, which prevents an intuitive understanding of its meaning. The unveiled relation to a robust MDP can allow us to understand what it means to minimize the value of an iterated risk measure in terms of uncertainties. In addition, the optimal policy for a robust MDP is often found too conservative [3, 4, 10, 21], or it becomes intractable to find the optimal policy particularly when the transition probabilities have uncertainties [3, 4, 10]. The unveiled relations to a risk-sensitive MDP, for which the optimal policy can be found efficiently, can allow us to find the optimal robust policy efficiently, avoiding that the policy is too conservative. We will explore these possibilities.

The contributions of this paper can be summarized in two points. First, we prove that a risk-sensitive MDP with the objective of minimizing the value of an iterated risk measure is equivalent to a robust MDP with the objective of minimizing the expected cumulative cost for the worst case when the probability mass functions for the transition and cost from a state have uncertainties. More specifically, the iterated risk measure of the risk-sensitive MDP is defined recursively with a class of coherent risk measures [9], and it evaluates the riskiness of the sum of the value of a coherent risk measure of immediate cost. The uncertainty set of the robust MDP is characterized by the use of a representation of the coherent risk measure. See Section 2.

Second, we prove that a risk-sensitive MDP with the objective of minimizing an expected exponential utility is equivalent to a robust MDP whose objective is to minimize the expected cumulative cost minus a penalty function for the worst case when the probability mass functions for the transition and cost from a state have uncertainties. More specifically, the expected exponential utility evaluates the riskiness of the sum of the value of an entropic risk measure [6] of immediate cost. The penalty function measures the deviation of the values of the probability mass functions from their nominal values using the Kullback-Leibler divergence. See Section 3.

## 2 Robust representations of iterated coherent risk measures

Throughout this paper, we consider Markov decision processes over a finite horizon, so that there are $N$ decision epochs. Let $\mathcal{S}_n$ be the set of possible states at the $n$-th decision epoch for $n = 0, \ldots, N-1$. Let $\mathcal{A}(s)$ be the set of candidate actions from the state, $s$. We assume that a nominal transition probability, $p_0(s'|s,a)$, is associated with the transition from each state $s \in \mathcal{S}_n$ to each state $s' \in \mathcal{S}_{n+1}$ given that the action $a \in \mathcal{A}(s)$ is taken at $s$ for $n = 0, \ldots, N-1$. For a robust MDP, the corresponding true transition probability, $p(s'|s,a)$, has the uncertainty that will be specified in the sequel. The random cost, $C(s,a)$, is associated with each pair of a state, $s$, and an action, $a \in \mathcal{S}(a)$. We assume that $C(s,a)$ has a nominal probability distribution, but the true probability distribution for a robust MDP has the uncertainty that will be specified in the sequel. We assume that $\mathcal{S}_i$ and $\mathcal{S}_j$ are disjoint for any $j \neq i$ (e.g., the state space is augmented with time).

### 2.1 Special case of the iterated conditional tail expectation

We start by studying a robust MDP where the uncertainty is specified by the factor, $\alpha$, such that $0 < \alpha < 1$, which determines the possible deviation from the nominal value. Specifically, for each pair of $s \in \mathcal{S}_n$ and $a \in \mathcal{A}(s)$, the true transition probabilities are in the following uncertainty set:

$$0 \leq p(s'|s,a) \leq \frac{1}{\alpha} p_0(s'|s,a), \forall s' \in \mathcal{S}_{n+1} \text{ and } \sum_{s' \in \mathcal{S}_{n+1}} p(s'|s,a) = 1. \tag{1}$$

Throughout Section 2.1, we assume that the cost $C(s,a)$ is deterministic and has no uncertainty.

Because the uncertainty set (1) is convex, the existing technique [11] can be used to efficiently find the optimal policy that minimizes the expected cumulative cost for the worst case where the true probability is chosen to maximize the expected cumulative cost within the uncertainty set:

$$\min_{\pi} \max_{p \in \mathcal{U}_p} \mathsf{E}_p[\tilde{C}(\pi)], \tag{2}$$

where $\tilde{C}(\pi)$ is the cumulative cost with a policy $\pi$, and $\mathsf{E}_p$ is the expectation with respect to $p$, which is chosen from the uncertainty set, $\mathcal{U}_p$, defined with (1).

Our key finding is that there is a risk-sensitive MDP that is equivalent to the robust MDP having the objective (2). Specifically, consider the risk-sensitive MDP, where the transition probability is given by $p_0$, and the cost $C(s,a)$ is deterministic given $s$ and $a$. This risk-sensitive MDP becomes equivalent to the robust MDP having the objective (2) when the objective of the risk-sensitive MDP is to find the optimal $\pi$ with respect to an iterated conditional tail expectation (ICTE) [13]:

$$\min_{\pi} \mathsf{ICTE}_\alpha^{(N)}[\tilde{C}(\pi)], \tag{3}$$

where $\mathsf{ICTE}_\alpha^{(N)}$ denotes the ICTE defined for $N$ decision epochs with parameter $\alpha$. Specifically, $\mathsf{ICTE}_\alpha^{(N)}$ is defined recursively with conditional tail expectation (CTE) as follows [13]:

$$\mathsf{ICTE}_\alpha^{(N-i+1)}\left[\tilde{C}(\pi)\right] \equiv \mathsf{CTE}_\alpha\left[\mathsf{ICTE}_\alpha^{(N-i)}\left[\tilde{C}(\pi)|S_i\right]\right], \text{ for } i = 1, \ldots, N, \tag{4}$$

$$\tag{5}$$

where we define $\mathsf{ICTE}_\alpha^{(0)}[\tilde{C}(\pi)] \equiv \tilde{C}(\pi)$. In (4), $\mathsf{ICTE}_\alpha^{(N-i)}[\tilde{C}(\pi)|S_i]$ denotes the ICTE of $\tilde{C}(\pi)$ conditioned on the state at the $i$-th decision epoch. When $S_i$ is random, so is $\mathsf{ICTE}_\alpha^{(N-i)}[\tilde{C}(\pi)|S_i]$. The right-hand side of (4) evaluates the CTE of this random $\mathsf{ICTE}_\alpha^{(N-i)}[\tilde{C}(\pi)|S_i]$. CTE is also known as conditional value at risk or average value at risk and is formally defined as follows for a random variable $Y$:

$$\mathsf{CTE}_\alpha[Y] \equiv \frac{(1-\beta)\mathsf{E}[Y|Y > V_\alpha] + (\beta - \alpha)V_\alpha}{1-\alpha}, \tag{6}$$

where $V_\alpha \equiv \min\{y \mid F_Y(y) \geq \alpha\}$, and $F_Y$ is the cumulative distribution function of $Y$. For a continuous $Y$, or unless there is a mass probability at $V_\alpha$, we have $\mathsf{CTE}_\alpha[Y] = \mathsf{E}[Y|Y > V_\alpha]$.

The equivalence between the robust MDP with the objective (2) and the risk-sensitive MDP with the objective (3) can be shown by the use of the following alternative definition of CTE:

$$\mathsf{CTE}_\alpha[Y] \equiv \max_{q \in \mathcal{Q}} \mathsf{E}_q[Y], \tag{7}$$

where $\mathcal{Q}$ is the set of probability mass functions, $q$, whose support is a subset of the probability mass function, $q_0$, of $Y$ such that $q(y) \leq q_0(y)/\alpha$ for every $y$ in the support of $q_0$. Specifically, let $C_i^\pi$ be the cost incurred at the $i$-th epoch with policy $\pi$ so that $\tilde{C}(\pi) = C_0^\pi + \cdots + C_{N-1}^\pi$. Then, by the recursive definition of ICTE and the translation invariance[1] of CTE, it can be shown that

$$\mathsf{ICTE}_\alpha^{(N-i+1)}\left[\tilde{C}(\pi) \mid S_{i-1}\right]$$

$$= \sum_{j=0}^{i-1} C_i^\pi + \mathsf{CTE}_\alpha\left[C_i^\pi + \mathsf{ICTE}_\alpha^{(N-i)}\left[\sum_{j=i+1}^{N-1} C_j^\pi \mid S_i\right] \mid S_{i-1}\right] \tag{8}$$

$$= \sum_{j=0}^{i-1} C_i^\pi + \max_{p \in \mathcal{U}_p} \mathsf{E}_p\left[C_i^\pi + \mathsf{ICTE}_\alpha^{(N-i)}\left[\sum_{j=i+1}^{N-1} C_j^\pi \mid S_i\right] \mid S_{i-1}\right], \tag{9}$$

where the second equality follows from (7). What (9) suggests is that the ICTE of the cumulative cost given $S_i$ can be represented by the cost already accumulated $C_0^\pi + \cdots + C_{i-1}^\pi$ plus the maximum possible expected value of the sum of the cost incurred at $S_i$ and the ICTE of the cumulative cost to be incurred from the $(i + 1)$st epoch. Induction can now be used to establish the following theorem, which will be proved formally for the general case in Section 2.3:

**Theorem 1.** *When the immediate cost from a state is deterministic given that state and the action from that state, the risk-sensitive MDP with the objective (3) is equivalent to the robust MDP with the objective (2).*

Throughout, we say that a risk-sensitive MDP is equivalent to a robust MDP if the two MDPs have a common state space, and, regardless of the values of the parameters of the MDPs, the optimal action for one MDP coincides with that for the other for every state.

## 2.2 Relation between cost uncertainty and risk-sensitivity

In addition to the transition probabilities, we now assume that the probability distribution of cost has uncertainty. Specifically, for each pair of $s \in \mathcal{S}_n$ and $a \in \mathcal{A}(s)$, the true probability mass function[2], $f(\cdot|s, a)$, for the random cost, $C(s, a)$, is in the following uncertainty set that is characterized with the nominal probability mass function, $f_0(\cdot|s, a)$:

$$0 \leq f(x|s, a) \leq \frac{1}{\alpha} f_0(x|s, a), \forall x \in \mathcal{X} \text{ and } \sum_{s \in \mathcal{X}(s,a)} f(x|s, a) = 1, \tag{10}$$

where $\mathcal{X}(s, a)$ is the support of $C(s, a)$. Because the uncertainty sets, (1) and (10), are both convex, the existing technique [11] can still be used to efficiently find the optimal policy with respect to

$$\min_\pi \max_{p \in \mathcal{U}_p, f \in \mathcal{U}_f} \mathsf{E}_{p,f}[\tilde{C}(\pi)], \tag{11}$$

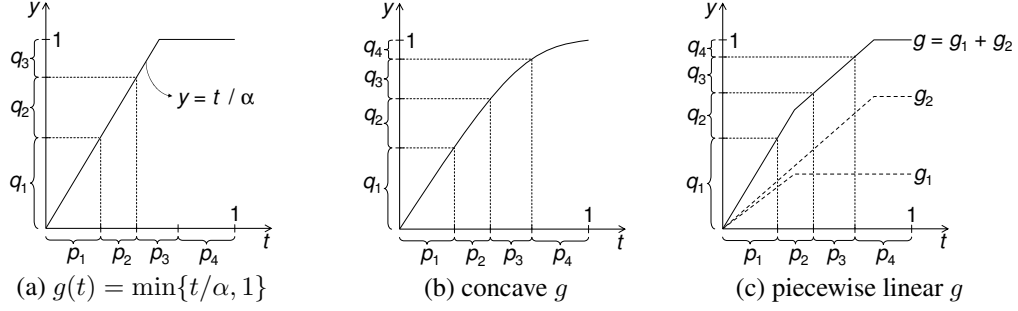

(a) $g(t) = \min\{t/\alpha, 1\}$      (b) concave $g$      (c) piecewise linear $g$

Figure 1: An illustration of the probabilities that give the worst case expectation.

where $\mathcal{U}_f$ is defined analogously to $\mathcal{U}_p$.

Again, our key finding is that there is a risk-sensitive MDP that is equivalent to the robust MDP having the objective (11). To define the objective of the equivalent risk-sensitive MDP, let $D(s, a) \equiv \mathsf{CTE}_\alpha[C(s, a)]$ and let $\tilde{D}(\pi)$ be the cumulative value of $D(s, a)$ along the sequence of $(s, a)$ with a policy $\pi$. Then the objective of the equivalent risk-sensitive MDP is given by

$$\min_\pi \mathsf{ICTE}_\alpha^{(N)}\left[\tilde{D}(\pi)\right]. \tag{12}$$

By first applying (7) to $D(s, a)$ and following the arguments that have led to Theorem 1, we can establish the following theorem, which will be proved formally for the general case in Section 2.3:

**Theorem 2.** *The risk-sensitive MDP with the objective (12) is equivalent to the robust MDP with the objective (11).*

## 2.3 General case of coherent risk measures

The robust MDPs considered in Section 2.1 and Section 2.2 are not quite flexible, can lead to too conservative policies depending on the value of $\alpha$, or might be too sensitive to the particular value of $\alpha$. We now introduce a broader class of robust MDPs and equivalent risk-sensitive MDPs.

To define the broader class of robust MDPs, we study the uncertainty set of (1) and (10) in more detail. Given a random variable that takes value $v_i$ with nominal probability $p_i$ for $i = 1, \ldots, m$, a step of finding the optimal robust policy calculates the maximum possible expected value:

$$
\begin{aligned}
\max_{\mathbf{q}} \quad & q_1 v_1 + \cdots + q_m v_m \\
\text{s.t.} \quad & 0 \le q_i \le \frac{1}{\alpha} p_i, \forall i = 1, \ldots m \\
& q_1 + \cdots + q_m = 1.
\end{aligned}
\tag{13}
$$

Without loss of generality, let $v_1 > v_2 > \ldots > v_m$. Then the optimal solution to (13) can be illustrated with Figure 1(a): for $i = 1, \ldots, m$, the optimal solution $\mathbf{q} \equiv (q_1, \ldots, q_m)$ satisfies

$$\sum_{\ell=1}^{i} q_\ell = g\left(\sum_{\ell=1}^{i} p_\ell\right), \tag{14}$$

where $g(t) = \min\{t/\alpha, 1\}$. Relaxing the constraints in (13), we obtain the following optimization problem, whose optimal solution is still given by (14):

$$
\begin{aligned}
\max_{\mathbf{q}} \quad & q_1 v_1 + \cdots + q_m v_m \\
\text{s.t.} \quad & \sum_{\ell=1}^{i} q_\ell \le g\left(\sum_{\ell=1}^{i} p_\ell\right), \forall i = 1, \ldots, m \\
& 0 \le q_i, \forall i = 1, \ldots, m.
\end{aligned}
\tag{15}
$$

The inflexibility of (15) stems from the inflexibility of $g(t) = \min\{t/\alpha, 1\}$, which has only one adjustable parameter, $\alpha$. When $\alpha$ is small (specifically, $0 < \alpha \le 1 - p_m$), some of the $q_i$s become

zero. This means that the corresponding optimistic cases (those resulting in small $v_i$s) are ignored. Otherwise (specifically, $1-p_m < \alpha \leq 1$), the uncertainty set can become too small as $q_i \leq p_i/\alpha, \forall i$.

This inflexibility motivates us to generalize $g$ to a concave function such that $g(0) = 0$ and $g(1) = 1$ (see Figure 1(b)). The optimal solution to (15) with the concave $g$ is still given by (14). With an appropriate $g$, we can consider a sufficiently large uncertainty set for the pessimistic cases (e.g., $q_1 \gg p_1$) and at the same time consider the possibility of the optimistic cases (e.g. $q_m > 0$).

To formally define the uncertainty set for $p(s'|s,a)$, $s \in \mathcal{S}_n$ and $a \in \mathcal{A}(s)$, with the concave $g$, let $Q_{p/p_0}(\cdot)$ denote the quantile function of a random variable that takes value $p(s'|s,a)/p_0(s'|s,a)$ with probability $p_0(s'|s,a)$ for $s' \in \mathcal{S}_{n+1}$. Analogously, let $Q_{f/f_0}(\cdot)$ denote the quantile function of a random variable that takes value $f(x|s,a)/f_0(x|s,a)$ with probability $f_0(x|s,a)$ for $x \in \mathcal{X}(s,a)$. Then $p(s'|s,a)$ and $f(x|s,a)$ are in the uncertainty set iff we have, for $0 < t < 1$, that

$$\int_{1-t}^{1} Q_{p/p_0}(u)\,du \leq g(t) \quad \text{and} \quad \int_{1-t}^{1} Q_{f/f_0}(u)\,du \leq g(t). \tag{16}$$

Now (7) suggests that expectation with respect to the $\mathbf{q}$ illustrated in Figure 1(a) is the CTE with parameter $\alpha$ with respect to the corresponding $\mathbf{p}$. It can be shown that the expectation with respect to the $\mathbf{q}$ illustrated in Figure 1(b) is a coherent risk measure, CRM, of the following form [9]:

$$\mathsf{CRM}_H[Y] = \int_0^1 \mathsf{CTE}_\alpha[Y]\,dH(\alpha), \tag{17}$$

for a nondecreasing function $H$ such that $H(0) = 0$ and $H(1) = 1$, where $Y$ denotes a generic random variable. Notice that (17) is a weighted average of $\mathsf{CTE}_\alpha[Y]$ for varying $\alpha$s. One can balance the weights on worse cases (higher $\alpha$) and the weights on better cases (lower $\alpha$).

Let $K(s,a) \equiv \mathsf{CRM}_H[C(s,a)]$ and let $\tilde{K}(\pi)$ be the cumulative value of $K(s,a)$ along the sequence of $(s,a)$ with a policy, $\pi$. We define an iterated coherent risk measure (ICRM) of $\tilde{K}(\pi)$ as follows:

$$\mathsf{ICRM}_H^{(N-i+1)}\left[\tilde{K}(\pi)\right] \equiv \mathsf{CRM}_H\left[\mathsf{ICRM}_H^{(N-i)}\left[\tilde{K}(\pi)|S_i\right]\right], \text{ for } i = 1,\ldots,N, \tag{18}$$

where $\mathsf{ICRM}_H^{(0)}[\tilde{K}(\pi)] \equiv \tilde{K}(\pi)$. Now we are ready to prove the general results in this section.

**Theorem 3.** *Consider the risk-sensitive MDP with the following objective:*

$$\min_\pi \mathsf{ICRM}_H^{(N)}\left[\tilde{K}(\pi)\right]. \tag{19}$$

*This risk-sensitive MDP is equivalent to the robust MDP with the objective (11) if*

$$\frac{dg(t)}{dt} = \int_t^1 \frac{1}{s}\,dH(s) \quad for \quad 0 < t < 1. \tag{20}$$

To gain an intuition behind (20), consider the $g$ illustrated in Figure 1(c), where $g_1(t) = \min\{x/\alpha_1, r_1\}$, $g_2(t) = \min\{x/\alpha_2, r_2\}$, and $g(t) = g_1(t) + g_2(t)$ for $0 \leq t \leq 1$. The expectation with respect to the $\mathbf{q}$ illustrated in Figure 1(c) can be represented by $r_1\mathsf{CRM}_{\alpha_1}[\cdot] + r_2\mathsf{CRM}_{\alpha_2}[\cdot]$ with respect to the corresponding $\mathbf{p}$. The $H$ is thus a piecewise constant function with a step of size $r_i$ at $\alpha_i$ for $i = 1, 2$. The slope $\frac{dg(t)}{dt}$ is either $1/\alpha_1 + 1/\alpha_2$, $1/\alpha_2$, or 0, depending on the particular value of $t$ in such a way that (20) holds.

*Proof of Theorem 3.* Notice that Bellman's optimality equations are satisfied both for the robust MDP and for the risk-sensitive MDP under consideration. For the robust MDP, Bellman's optimality equations are established in [11]. For our risk-sensitive MDP, note that the coherent risk measure satisfies strong monotonicity, translation-invariance, and positive homogeneity that are used to establish Bellman's optimality equations in [13]. A difference between the risk-sensitive MDP in [13] and our risk-sensitive MDP is that the former minimizes the value of an iterated risk measure for $\tilde{C}$, while the latter minimizes the value of an iterated risk measure (specifically, $\mathsf{ICRM}_H^{(0)}$) for $\tilde{K}$. This difference does not affect whether Bellman's optimality equations are satisfied.

The equivalence between our risk-sensitive MDP and our robust MDP can thus be established by showing that the two sets of Bellman's optimality equations are equivalent. For $s \in \mathcal{S}_n$, Bellman's optimality equation for our robust MDP is

$$v(s) \quad = \quad \min_{a \in \mathcal{A}(s)} \max_{\mathbf{p} \in \mathcal{U}_p, \mathbf{f} \in \mathcal{U}_f} \left( \sum_{x \in \mathcal{X}(s,a)} x \, f(x|s,a) + \sum_{s' \in \mathcal{S}_{n+1}} v(s') \, p(s'|s,a) \right), \qquad (21)$$

where $v(s)$ denotes the value function representing the worst-case expected cumulative cost from $s$. For $s \in \mathcal{S}_n$, Bellman's optimality equation for our risk-sensitive MDP is

$$w(s) \quad = \quad \min_{a \in \mathcal{A}(s)} \mathsf{CRM}_H \left[ \mathsf{CRM}_H[C_0(s,a)] + W(s,a) \right], \qquad (22)$$

where $w(s)$ denotes the value function representing the value of the iterated coherent risk measure from $s$, $C_0(s,a)$ is a random variable that takes value $x$ with probability $f_0(x|s,a)$ for $x \in \mathcal{X}(s,a)$, and $W(s,a)$ is a random variable that takes value $w(s')$ with probability $p_0(s'|s,a)$ for $s' \in \mathcal{S}_{n+1}$. Note that the inner $\mathsf{CRM}_H$ is calculated with respect to $f_0(\cdot|s,a)$; the outer $\mathsf{CRM}_H$ is with respect to $p_0(\cdot|s,a)$. In the following, we will show the equivalence between (21) and (22).

We first show the following equality:

$$\max_{\mathbf{f} \in \mathcal{U}_f} \sum_{x \in \mathcal{X}(s,a)} x \, f(x|s,a) \quad = \quad \mathsf{CRM}_H[C_0(s,a)]. \qquad (23)$$

Let $\{x_1, x_2, \ldots\} = \mathcal{X}(s,a)$ such that $x_1 > x_2 > \ldots$. As we have seen with Figure 1, the maximizer, $f^\star$, of the left-hand side of (23) satisfies

$$\sum_{\ell=1}^{i} f^\star(x_\ell|s,a) \quad = \quad g\left( \sum_{\ell=1}^{i} f_0(x_\ell|s,a) \right) \qquad (24)$$

for $i = 1, 2, \ldots$. For brevity, let $R_i \equiv \sum_{\ell=1}^{i} f_0(x_\ell|s,a)$. Then we have

$$\max_{\mathbf{f} \in \mathcal{U}_f} \sum_{x \in \mathcal{X}(s,a)} x \, f(x|s,a) = \sum_i x_i \int_{R_{i-1}}^{R_i} dg(t) = \sum_i x_i \int_{R_{i-1}}^{R_i} \int_t^1 \frac{1}{u} \, dH(u) \, dt, \qquad (25)$$

where the first equality follows from (24), and the second equality follows from (20). Exchanging the integrals in the last expression, we obtain

$$\max_{\mathbf{f} \in \mathcal{U}_f} \sum_{x \in \mathcal{X}(s,a)} x \, f(x|s,a) \quad = \quad \sum_i \int_{R_{i-1}}^{1} x_i \int_{R_{i-1}}^{\min\{u, R_i\}} dt \, \frac{1}{u} \, dH(u) \qquad (26)$$

$$= \quad \sum_i \int_{R_{i-1}}^{1} x_i \, (\min\{u, R_i\} - R_{i-1}) \, \frac{1}{u} \, dH(u). \qquad (27)$$

Exchanging the integral and the summation in the last expression, we obtain

$$\max_{\mathbf{f} \in \mathcal{U}_f} \sum_{x \in \mathcal{X}(s,a)} x \, f(x|s,a) \quad = \quad \int_0^1 \frac{\sum_{i:R_{i-1} \leq u} x_i \, (\min\{u, R_i\} - R_{i-1})}{u} \, dH(u) \qquad (28)$$

$$= \quad \int_0^1 \mathsf{CTE}_u[C_0(s,a)] \, dH(u), \qquad (29)$$

which establishes (23). To understand the last equality, plug in the following expressions in (6): $\alpha = 1 - u$, $V_\alpha = x_{i^\star}$, and $\beta = 1 - R_{i^\star - 1}$, where $i^\star \equiv \min\{i | R_i > u\}$.

Finally, we show the equivalence between (21) and (22). By (23), we have

$$\max_{\mathbf{p} \in \mathcal{U}_p, \mathbf{f} \in \mathcal{U}_f} \left( \sum_{x \in \mathcal{X}(s,a)} x \, f(x|s,a) + \sum_{s' \in \mathcal{S}_{n+1}} v(s') \, p(s'|s,a) \right) = \mathsf{CRM}_H[C_0(s,a)] + \max_{\mathbf{p} \in \mathcal{U}_p} \sum_{s' \in \mathcal{S}_{n+1}} v(s') \, p(s'|s,a). \quad (30)$$

Analogously to (23), we can show

$$\max_{\mathbf{p} \in \mathcal{U}_p} \sum_{s' \in \mathcal{S}_{n+1}} v(s') \, p(s'|s,a) \quad = \quad \mathsf{CRM}_H[V_0(s,a)], \qquad (31)$$

where $V_0(s, a)$ denotes the random variable that takes value $v(s')$ with probability $p_0(s'|s, a)$ for $s' \in \mathcal{S}_{n+1}$. By (30) and (31), we have

$$v(s) = \min_{a \in \mathcal{A}(s)} \left( \mathsf{CRM}_H[C_0(s, a)] + \mathsf{CRM}_H[V(s, a)] \right), \tag{32}$$

where the first $\mathsf{CTE}_H$ is with respect to $f_0(\cdot|s, a)$; the second is with respect to $p_0(\cdot|s, a)$. Because $f_0(\cdot|s, a)$ is independent of the state at $n + 1$, the translation invariance[3] of $\mathsf{CRM}_H$ implies

$$v(s) = \min_{a \in \mathcal{A}(s)} \mathsf{CRM}_H[\mathsf{CRM}_H[C_0(s, a)] + V(s, a)], \tag{33}$$

where the inner $\mathsf{CRM}_H$ is with respect to $f_0(\cdot|s, a)$; the outer is with respect to $p_0(\cdot|s, a)$. Because $v(s') = w(s') = 0, \forall s' \in \mathcal{S}_N$, we can show, by induction, that (33) is equivalent to (22). $\qquad\square$

# 3 Robust representations of expected exponential utilities

In this section, we study risk-sensitive MDPs whose objectives are defined with expected exponential utilities. We will see that there are robust MDPs that are equivalent to these risk-sensitive MDPs.

We start by the standard risk-sensitive MDP [5, 7, 8, 15, 18] whose objective is to minimize $\mathsf{E}[\exp(\gamma \tilde{C}(\pi))]$ for $\gamma > 0$. Because $\gamma > 0$, minimizing $\mathsf{E}[\exp(\gamma \tilde{C}(\pi))]$ is equivalent to minimizing an entropic risk measure (ERM) [6, 13]: $\mathsf{ERM}_\gamma[\tilde{C}(\pi)] \equiv \frac{1}{\gamma} \ln \mathsf{E}[\exp(\gamma \tilde{C}(\pi))]$.

The key property of ERM that we exploit in this section is

$$\mathsf{ERM}_\gamma[Y] = \max_{q \in \mathcal{P}(q_0)} \left\{ \mathsf{E}_{q_0}[Y] - \gamma \, \mathrm{KL}(q||q_0) \right\}, \tag{34}$$

where $Y$ is a generic discrete random variable, $q_0$ is the probability mass function for $Y$, $\mathcal{P}(q_0)$ is the set of probability mass functions whose support is contained in the support of $q_0$ (i.e., $q(y) = 0$ if $q_0(y) = 0$ for $q \in \mathcal{P}(q_0)$), $\mathsf{E}_q$ is the expectation with respect to $q \in \mathcal{P}(q_0)$, and $\mathrm{KL}(q||q_0)$ is the Kullback-Leibler divergence [2] from $q$ to $q_0$. The property (34) has been discussed in the context of optimal control [1, 16]. See [6] for a proof of (34). Observe that the maximizer of the right-hand side of (34) trades a large value of $\mathsf{E}_q[Y]$ for the closeness of $q$ to $q_0$.

It is now evident that the risk-sensitive MDP with the objective of minimizing $\mathsf{E}[\exp(\gamma \tilde{C}(\pi))]$ is equivalent to a "robust" MDP with the objective of minimizing $\mathsf{E}_q[\tilde{C}(\pi)] - \gamma \, \mathrm{KL}(q||q_0)$ for the worst choice of $q \in \mathcal{P}(q_0)$, where $q_0$ denotes the probability mass function for $\tilde{C}(\pi)$. Here, the uncertainty is in the distribution of the cumulative cost, and it is nontrivial how this uncertainty is related to the uncertainty in the parameters, $p$ and $f$, of the MDP.

Our goal is to explicitly relate the risk-sensitive MDP of minimizing $\mathsf{E}[\exp(\gamma \tilde{C}(\pi))]$ to uncertainties in the parameters of the MDP. For a moment, we assume that $C(s, a)$ has no uncertainty and is deterministic given $s$ and $a$, which will be relaxed later.

To see the relation, we study $\mathsf{ERM}_\gamma[\tilde{C}(\pi)]$ for a given $\pi$. Let $p_0^\pi(s_{i+1}|s_i)$ be the nominal transition probability from $s_i \in \mathcal{S}_i$ to $s_{i+1} \in \mathcal{S}_{i+1}$ for $i = 0, \ldots, N - 1$. By the translation invariance and the recursiveness[4] of ERM [13], we have

$$\mathsf{ERM}_\gamma[\tilde{C}(\pi)] = C_0^\pi + \mathsf{ERM}_\gamma \left[ C_1^\pi + \mathsf{ERM}_\gamma \left[ \sum_{i=2}^{N-1} C_i^\pi | S_1 \right] \right], \tag{35}$$

where the inner ERM is with respect to $p_0^\pi(\cdot|S_1)$; the outer is with respect to $p_0^\pi(\cdot|s_0)$. By (34), the second term of the right-hand side of (35) can be represented as follows:

$$\max_{p^\pi(\cdot|s_0) \in \mathcal{P}(p_0^\pi(\cdot|s_0))} \left\{ \mathsf{E}_{p^\pi(\cdot|s_0)} \left[ C_1^\pi + \mathsf{ERM}_\gamma \left[ \sum_{i=2}^{N-1} C_i^\pi \mid S_1 \right] \right] - \gamma \, \mathrm{KL} \left( p^\pi(\cdot|s_0) || p_0^\pi(\cdot|s_0) \right) \right\} \tag{36}$$

Thus, by induction, we can establish the following theorem[5]:

**Theorem 4.** *When the immediate cost from a state is deterministic given that state and the action from that state, the risk-sensitive MDP with the objective of minimizing* $\mathsf{E}[\exp(\gamma \tilde{C}(\pi))]$ *is equivalent to the robust MDP with the following objective:*

$$\min_{\pi} \max_{p^{\pi} \in \mathcal{P}(p_0^{\pi})} \left\{ \mathsf{E}_{p^{\pi}} \left[ \sum_{i=0}^{N-1} C_i^{\pi} \right] - \gamma \, \mathsf{E}_{p^{\pi}} \left[ \sum_{i=0}^{N-2} \mathrm{KL}\left(p^{\pi}(\cdot|S_i) || p_0^{\pi}(\cdot|S_i)\right) \right] \right\}, \tag{37}$$

*where* $p^{\pi} \in \mathcal{P}(p_0^{\pi})$ *denotes that* $p^{\pi}(\cdot|s_i) \in \mathcal{P}_{p_0^{\pi}(\cdot|s_i)}, \forall s_i \in \mathcal{S}_i, i = 0, \dots, N-1.$

Our results in Section 2.2 motivate us to extend Theorem 4 to the case where $C(s, a)$ has uncertainties. Let $f_0^{\pi}(\cdot|s)$ be the nominal probability mass function for the immediate cost from a state $s$ under a policy $\pi$. Consider the risk-sensitive MDP with the following objective:

$$\min_{\pi} \mathsf{ERM}_{\gamma} \left[ \tilde{L}(\pi) \right], \tag{38}$$

where $\tilde{L}(\pi)$ is the cumulative value of $L(s, a) \equiv \mathsf{ERM}_{\gamma}[C(s, a)]$ along the sequence of $(s, a)$ with a policy $\pi$. Then we have the following theorem, which is proved in the supplementary material.

**Theorem 5.** *The risk-sensitive MDP with the objective (38) is equivalent to the robust MDP with the following objective, where* $f^{\pi} \in \mathcal{P}(f_0^{\pi})$ *is defined analogously to* $p^{\pi} \in \mathcal{P}(p_0^{\pi})$ *in Theorem 4:*

$$\min_{\substack{\pi \\ p^{\pi} \in \mathcal{P}(p_0^{\pi}) \\ f^{\pi} \in \mathcal{P}(f_0^{\pi})}} \max \left\{ \begin{array}{l} \mathsf{E}_{p^{\pi}, f^{\pi}} \left[ \sum_{i=0}^{N-1} C_i^{\pi} \right] \\ -\gamma \, \mathsf{E}_{p^{\pi}} \left[ \sum_{i=0}^{N-2} \mathrm{KL}\left(p^{\pi}(\cdot|S_i) || p_0^{\pi}(\cdot|S_i)\right) + \sum_{i=0}^{N-1} \mathrm{KL}\left(f^{\pi}(\cdot|S_i) || f_0^{\pi}(\cdot|S_i)\right) \right] \end{array} \right\}. \tag{39}$$

# 4 Conclusion

We have shown relations between risk-sensitive MDPs and robust MDPs. Because ERM is also an iterated risk measure [13], the objectives of the risk-sensitive MDPs studied in this paper are all with respect to some iterated risk measures. The significance of iterated risk measures is intensively discussed in [13], but it can represent one's preferences that cannot be represented by standard expected exponential utility and yet allows efficient optimization and consistent decision making. While the prior work [13, 14, 19] minimizes the iterated risk measure of the cumulative cost ($\tilde{C}(\pi)$ in Section 2), our study on the relation to a robust MDP suggests that one might want to minimize the iterated risk measure of the sum of the values of risk measures for immediate costs (e.g., $\tilde{K}(\pi)$ in Section 2.3 or $\tilde{L}(\pi)$ in Section 3), because the latter is related to the robustness against uncertainty in cost. The optimal policy with respect to an iterated risk measure can be found efficiently with dynamic programming (specifically, the computational effort that is required in addition to that of the dynamic programming for minimizing the expected cumulative cost is in the time to calculate a risk measure such as CTE instead of expectation at each step of the dynamic programming) [13]. This means that the optimal policy for the robust MDP studied in this paper can be found quite efficiently. In particular, the robust MDP in Theorem 5 might not seem to allow an efficient optimization without the knowledge of the relation to the corresponding risk-sensitive MDP, whose optimal policy is readily available with dynamic programming. Overall, the relation to a robust MDP can provide strong motivation for the corresponding risk-sensitive MDP and vice versa.

For simplicity, the uncertainty sets in Section 2 are characterized by a single parameter, $\alpha$, or a single function, $g$, but it is trivial to extend our results to the cases where the uncertainty sets are defined differently depending on the particular states, actions, and other elements of the MDP. In such cases, the objective of the corresponding risk-sensitive MDP is composed of various risk measures. The uncertainty set in Section 3 depends only on the support of the nominal probability mass function. The penalty for the deviation from the nominal value can be adjusted with a single parameter, $\gamma$, but it is also trivial to extend our results to the cases, where this parameter varies depending on the particular elements of the MDP. In such cases, the objective of the corresponding risk-sensitive MDP is an iterated risk measure composed of ERM having varying parameters. It would also be an interesting direction to extend our results to convex risk measures, which allows robust representations.

## Footnotes

[1] $\mathsf{CTE}_\alpha[Y + b] = \mathsf{CTE}_\alpha[Y] + b$ for a random $Y$ and a deterministic $b$.

[2] Continuous cost is discussed in the supplementary material.

[3] $\mathsf{CRM}_H[Y + c] = \mathsf{CRM}_H[Y] + c$ for a deterministic constant $c$.

[4] $\mathsf{ERM}_\gamma[Y + c] = \mathsf{ERM}_\gamma[Y] + c$ and $\mathsf{ERM}_\gamma[Y] = \mathsf{ERM}_\gamma[\mathsf{ERM}_\gamma[Y|Z]]$, where $Y$ and $Z$ are generic random variables, and $c$ is a deterministic constant.

[5] The proof is omitted, because this is a spacial case of Theorem 5.

# References

[1] C. D. Charalambous, F. Rezaei, and A. Kyprianou. Relations between information theory, robustness, and statistical mechanics of stochastic systems. In *Proceedings of the 43rd IEEE Conference on Decision and Control*, volume 4, pages 3479–3484, 2004.

[2] T. M. Cover and J. A. Thomas. *Elements of Information Theory*. John Wiley & Sons, Inc., Hoboken, New Jersey, 2nd edition, 2006.

[3] E. Delage and S. Mannor. Percentile optimization in uncertain MDP with application to efficient exploration. In *Proceedings of the 24th Annual International Conference on Machine Learning (ICML 2007)*, pages 225–232, June 2007.

[4] E. Delage and S. Mannor. Percentile optimization for MDP with parameter uncertainty. *Operations Research*, 58(1):203–213, 2010.

[5] E. V. Denardo and U. G. Rothblum. Optimal stopping, exponential utility, and linear programming. *Mathematical Programming*, 16:228–244, 1979.

[6] H. Föllmer and A. Schied. *Stochastic Finance: An Introduction in Discrete Time*. Walter de Gruyter, Berlin, Germany, 3rd edition, 2010.

[7] R. Howard and J. Matheson. Risk-sensitive Markov decision processes. *Management Science*, 18(7):356–369, 1972.

[8] S. C. Jaquette. A utility criterion for Markov decision processes. *Management Science*, 23(1):43–49, 1976.

[9] S. Kusuoka. On law invariant coherent risk measures. In S. Kusuoka and T. Maruyama, editors, *Advances in Mathematical Economics*, volume 3, pages 83–95. Springer, Tokyo, 2001.

[10] S. Mannor, O. Mebel, and H. Xu. Lightning does not strike twice: Robust MDPs with coupled uncertainty. In *Proceedings of the International Conference on Machine Learning (ICML 2012)*, pages 385–392, 2012.

[11] A. Nilim and L. El Ghaoui. Robust control of Markov decision processes with uncertain transition matrices. *Operations Research*, 53(5):780–798, 2005.

[12] A. Nilim and L. E. Ghaoui. Robustness in Markov decision problems with uncertain transition matrices. In S. Thrun, L. Saul, and B. Schölkopf, editors, *Advances in Neural Information Processing Systems 16*. MIT Press, Cambridge, MA, 2004.

[13] T. Osogami. Iterated risk measures for risk-sensitive Markov decision processes with discounted cost. In *Proceedings of the 27th Conference on Uncertainty in Artificial Intelligence (UAI 2011)*, pages 567–574, July 2011.

[14] T. Osogami and T. Morimura. Time-consistency of optimization problems. In *Proceedings of the 26th Conference on Artificial Intelligence (AAAI-12)*, July 2012.

[15] S. D. Patek. On terminating Markov decision processes with a risk-averse objective function. *Automatica*, 37(9):1379–1386, 2001.

[16] I. R. Petersen, M. R. James, and P. Dupuis. Minimax optimal control of stochastic uncertain systems with relative entropy constraints. *IEEE Transactions on Automatic Control*, 45(3):398–412, 2000.

[17] M. L. Puterman. *Markov Decision Processes: Discrete Dynamic Programming*. Wiley-Interscience, Hoboken, NJ, second edition, 2005.

[18] U. G. Rothblum. Multiplicative Markov decision chains. *Mathematics of Operations Research*, 9(1):6–24, 1984.

[19] A. Ruszczyński. Risk-averse dynamic programming for Markov decision processes. *Mathematical Programming*, 125:235–261, 2010.

[20] H. Xu and S. Mannor. The robustness-performance tradeoff in Markov decision processes. In B. Schölkopf, J. Platt, and T. Hoffman, editors, *Advances in Neural Information Processing Systems 19*, pages 1537–1544. MIT Press, Cambridge, MA, 2007.

[21] H. Xu and S. Mannor. Distributionally robust Markov decision processes. In J. Lafferty, C. K. I. Williams, J. Shawe-Taylor, R. Zemel, and A. Culotta, editors, *Advances in Neural Information Processing Systems 23*, pages 2505–2513. MIT Press, Cambridge, MA, 2010.

